# PCA-Pyramids for Image Compression*

**Horst Bischof**
Department for Pattern Recognition
and Image Processing
Technical University Vienna
Treitlstraße 3/1832
A-1040 Vienna, Austria
bis@prip.tuwien.ac.at

**Kurt Hornik**
Institut für Statistik und
Wahrscheinlichkeitstheorie
Technische Universität Wien
Wiedner Hauptstraße 8–10/1071
A-1040 Vienna, Austria
Kurt.Hornik@ci.tuwien.ac.at

## Abstract

This paper presents a new method for image compression by neural networks. First, we show that we can use neural networks in a pyramidal framework, yielding the so-called PCA pyramids. Then we present an image compression method based on the PCA pyramid, which is similar to the Laplace pyramid and wavelet transform. Some experimental results with real images are reported. Finally, we present a method to combine the quantization step with the learning of the PCA pyramid.

## 1 Introduction

In the past few years, a lot of work has been done on using neural networks for image compression, cf. e.g. (Cottrell et al., 1987; Sanger, 1989; Mougeot et al., 1991; Schweizer et al., 1991)). Typically, networks which perform a Principal Component Analysis (PCA) were employed; for a recent overview of PCA networks, see (Baldi and Hornik, 1995).

A well studied and thoroughly understood PCA network architecture is the linear autoassociative network, see (Baldi and Hornik, 1989; Bourlard and Kamp, 1988). This network consists of $N$ input and output units and $M < N$ hidden units, and is

*This work was supported in part by a grant from the Austrian National Fonds zur Förderung der wissenschaftlichen Forschung (No. S7002MAT) to Horst Bischof.

trained (usually by back-propagation) to reproduce the input at the output units. All units are linear. Bourlard & Kamp (Bourlard and Kamp, 1988) have shown that at the minimum of the usual quadratic error function $\mathcal{E}$, the hidden units project the input on the space spanned by the first $M$ principal components of the input distribution. In fact, as long as the output units are linear, nothing is gained by using non-linear hidden units. On average, all hidden units have equal variance.

However, PCA is not the only method for image compression. Among many others, the Laplace Pyramid (Burt and Adelson, 1983) and wavelets (Mallat, 1989) have successfully been used to compress images. Of particular interest is the fact that these techniques provide a hierarchical representation of the image which can be used for progressive image transmission. However, these hierarchical methods are not adaptive.

In this paper, we present a combination of autoassociative networks with hierarchical methods. We propose the so-called PCA pyramids, which can be seen as an extension of image pyramids with a learning algorithm as well as cascaded locally connected autoassociative networks. In other words, we combine the structure of image pyramids and neural network learning algorithms, resulting in learning pyramids.

The structure of this paper is as follows. We first present image pyramids and, in particular, the PCA pyramid. Then, we discuss how these pyramids can be used for image compression, and present some experimental results. Next, we discuss a method to combine the quantization step of compression with the transformation. Finally, we give some conclusions and an outline of further research.

## 2   The PCA Pyramid

Before we introduce the PCA pyramid, let us describe regular image pyramids. For a discussion of irregular pyramids and their relation to neural networks, see (Bischof, 1993). In the simplest case, each successive level of the pyramid is obtained from the previous level by a filtering operation followed by a sampling operator. More general functions can be used to achieve the desired reduction. We therefore call them *reduction functions*. The structure of a pyramid is determined by the neighbor relations within the levels of the pyramid and by the "father-son" relations between adjacent levels. A cell (if it is not at the base level) has a set of *children (sons)* at the level directly below which provide input to the cell, a set of *neighbors (brothers/sisters)* at the same level, and (if it is not the apex of the pyramid) a set of *parents (fathers)* at the level directly above. We denote the structure of a (regular) pyramid by the expression $n \times n/r$, where $n \times n$ (the number of sons) is the size of the reduction window and $r$ the reduction factor which describes how the number of cells decreases from level to level.

### 2.1   PCA Pyramids

Since a pyramid reduces the information content of an image level by level, an objective for the reduction function would be to *preserve as much information as possible*, given the restrictions imposed by the structure of the pyramid, or equivalently, to minimize the information loss by the reduction function. This naturally

leads to the idea of representing the pyramid by a suitable PCA network. Among the many alternatives for such networks, we have chosen the autoassociative networks for two reasons. First, the analysis of Hornik & Kuan (Hornik and Kuan, 1992) shows that these networks are more stable than competing models. Second, autoassociative networks have the nice feature that they automatically provide us with the expansion function (weights from the hidden layer to output layer).

Since the neural network should have the same connectivity as the pyramid (i.e., the same father-son relations), its topology is determined by the structure of the pyramid. In this paper, we confine ourselves to the $4 \times 4/4$ pyramid for two reasons. First, the $4 \times 4/4$ pyramid has the nice property that every cell has the same number of fathers, which results in homogeneous networks. Second, as experiments have shown (Bischof, 1993) the results achieved with this pyramid are similar to other structures, e.g. the $5 \times 5/4$ pyramid, using fewer weights.

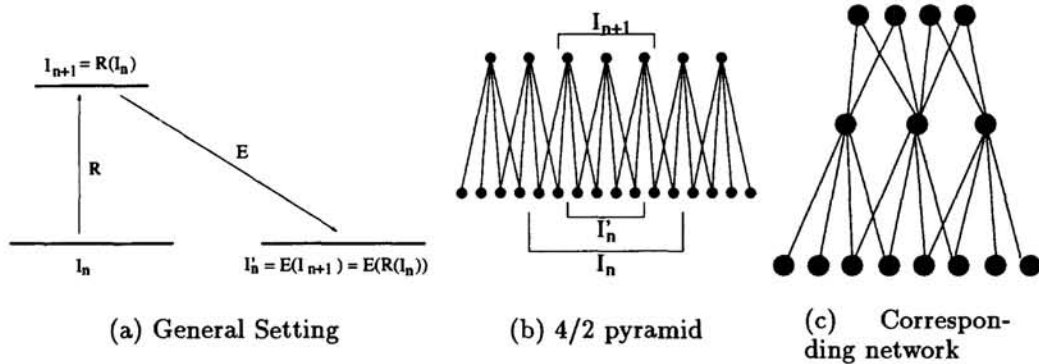

(a) General Setting    (b) 4/2 pyramid    (c) Corresponding network

Figure 1: From the structure of the pyramid to the topology of the network

Figure 1 depicts the one-dimensional situation of a 4/2 pyramid (this is the one-dimensional counterpart of the two-dimensional $4 \times 4/2$ pyramid). Figure 1a shows the general goal to be achieved and the notations employed; Figure 1b shows a 4/2 pyramid. When constructing the corresponding network, we start at the output layer (i.e., $I'_n$). For an $n/r$ pyramid we typically choose the size of the output layer as $n$. Next, we have to include all fathers of the cells in the output layer as hidden units. Finally, we have to include all sons of the hidden layer cells in the input layer. For the 4/2 pyramid, this results in an 8–3–4 network as shown in Figure 1c. A similar construction yields an $8 \times 8$–$3 \times 3$–$4 \times 4$ network for the $4 \times 4/4$ pyramid.

The next thing to consider are the constraints on the network weights due to the overlaps in the pyramid. To completely cover the input image with output units, we can shift the network only by four cells in each direction. Therefore, the hidden units at the borders overlap. For the 4/2 pyramid, the left and right hidden units must have identical weights. In the case of the $4 \times 4/4$ pyramid, the network has four independent units.

The thus constructed network can be trained by some suitable learning algorithm, typically of the back-propagation type, using batches of an image as input for trai-

ning the first pyramid level. After that, the second level of the pyramid can be trained in the same way using the first pyramid level as training data, and so on.

## 2.2   PCA-Laplace Pyramid and Image Compression

Thus far, we have introduced a network which can learn the reduction function $R$ and the expansion function $E$ of a pyramid. Analogously to the Laplace pyramid and the wavelet transform we can now introduce the *level $L_i$* of the PCA-Laplace pyramid, given by

$$L_i = I_i - I_i' = I_i - E(R(I_i))$$

It should be noted that during learning we exactly minimize the squared Laplace

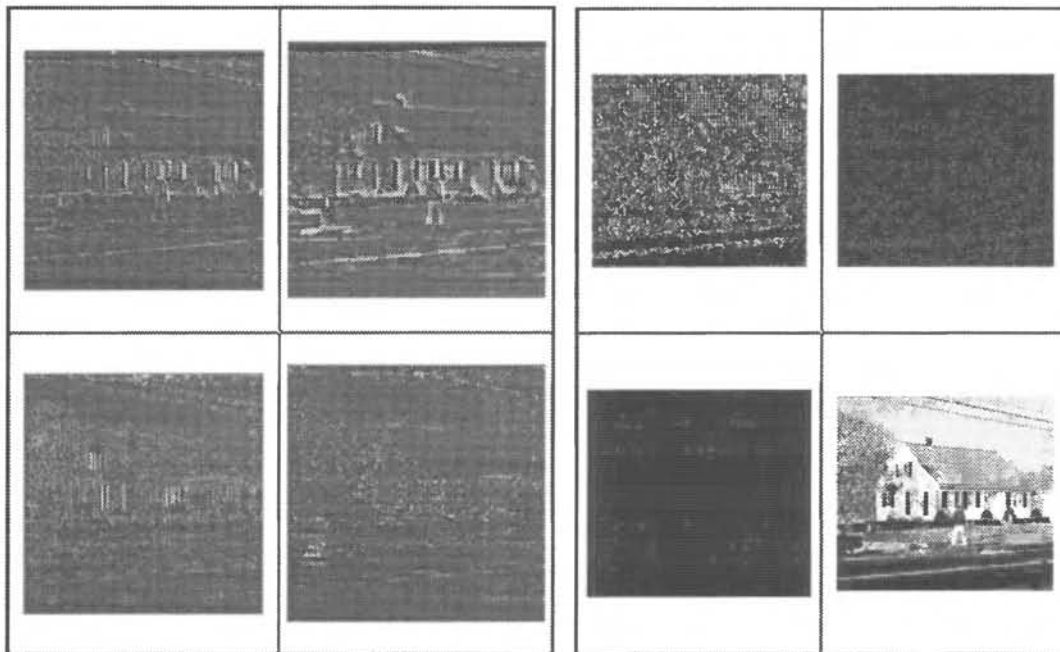

(a) First 2 levels of a Laplace pyramid (upper half) and PCA-Laplace pyramid (lower half) (grey = 0)

(b) Reconstruction error of house image with quantization of 3 bits, 4 bits, 7 bits, and reconstructed image

Figure 2: Results of PCA-Laplace-Pyramid

level. The original image $I_0$ can be completely recovered from level $I_n$ and the Laplace levels $L_0, \ldots, L_{n-1}$ by

$$I_0 = E(\cdots E(E(I_n) + L_{n-1}) + L_{n-2} \cdots) + L_0.$$

Since the level $I_n$ is rather small (e.g., $32 \times 32$ pixels) and the levels of the PCA-Laplace pyramid are typically sparse (i.e., many pixels are zero, see Figure 2a) and can therefore be compressed considerably by a conventional compression algorithm

(e.g. Lempel-Ziv (Ziv and Lempel, 1977)), this image representation results in a lossless image compression algorithm.

In order to achieve higher compression ratios we can quantize the levels of the PCA-Laplace pyramid. In this case, the compression is lossy, because the original image cannot be recovered exactly. The compression ratio and the amount of loss can be controlled by the number of bits used to quantize the levels of the PCA-Laplacian.

To measure the difference between the compressed and the original image, we use the normalized mean squared error (NMSE) as in (Cottrell et al., 1987; Sanger, 1989). The NMSE is given by the mean squared error divided by the average squared intensity of the image, i.e.,

$$\text{NMSE} = \frac{\text{MSE}}{\langle I_0^2 \rangle} = \frac{\langle (I_0 - C(I_0))^2 \rangle}{\langle I_0^2 \rangle},$$

where $I_0$ and $C(I_0)$ are the original and the compressed image, respectively. The compression ratio is measured by the amount of bits used to store $I_0$, divided by the amount of bits used to store $C(I_0)$.

## 2.3 Results

For the results reported here we trained the networks by a conjugate gradient algorithm for 100 steps[1] and used a uniform quantization which is fixed for all levels of the pyramid. As was shown in (Burt and Adelson, 1983; Mayer and Kropatsch, 1989), the results could be improved by gradually increasing the quantization from bottom to top.

Figure 2b shows the error images when the levels of the PCA-Laplacian pyramid are quantized with 3, 4, and 7 bits and the reconstructed image from the 7 bit Laplacian. Note that we used the same lookup-table for the error images. To compress the levels of the PCA-Laplacian pyramid, we employed the standard UNIX compress program which implements a Lempel-Ziv algorithm.

¿From these images one can see that the results with the 4 and 7 bit quantization are very good. Visually, no difference between the reconstructed and the original image can be perceived. Table 1 shows the compression ratios and the NMSEs on these images. We have performed experiments on 20 different images, the results on these images are comparable to the ones reported here.

These results compare favorably with the results in the literature (see Table 1). We have also applied a $5 \times 5/4$ Laplace pyramid to the house image which gave a compression ratio of 3.42 with an NMSE of 0.000087 for quantization with four bits of the Laplace levels. We have also included results achieved with JPEG. One can see that our method gives considerably better results.

We have also demonstrated experimentally what happens if we train a pyramid on one image and then apply this pyramid to another image without retraining. These experiments indicate that the errors are only a little bit larger for images not trained on. With five additional steps of training the errors are almost the same. ¿From

| Quant. | Compression ratio | Bits/Pixel | NMSE |
|---|---|---|---|
| 3 Bit | 37.628 | 0.212 | 0.0172 |
| 4 Bit | 24.773 | 0.323 | 0.0019 |
| 7 Bit | 8.245 | 0.970 | 0.0000215 |
| no Quant. | 3.511 | 2.279 | 0.0 |
| Cottrell (Cottrell et al., 1987) | 8.0 | 1.000 | 0.0059 |
| Sanger (Sanger, 1989) | 22.0 | 0.360 | 0.043 |
| $5 \times 5/4$ Laplace | 3.420 | 2.339 | 0.000087 |
| JPEG | 8.290 | 0.965 | 0.00139 |
| JPEG | 15.774 | 0.507 | 0.00348 |

Table 1: Compression ratios and NMSE for various compression methods

this results we can conclude that we do not need to retrain the pyramid for each new image.

## 3  Integration of Quantization

For the results reported in the previous section we have used a fixed and uniform quantization scheme which can be improved by using adaptive quantizers like the Lloyd I algorithm, Kohonen's Feature Maps, learning vector quantization, or something similar. Such an approach as taken by Schweizer (Schweizer et al., 1991) who combined a Cottrell-type network with self-organizing feature maps. However, we can go further.

With the PCA network we minimize the squared Laplace level which does not necessarily yield low compression errors. What we really want to minimize are the *quantized* Laplace levels. Usually, the Laplace levels have an unimodally shaped histogram centered at zero. However, for the result of the compression (i.e., compression ratio and NMSE), it is irrelevant if we shift the histogram to the left or the right as long as we shift the quantization intervals in the same way. The best results could be achieved if we have a multimodal histogram with peaks centered at the quantization points.

Using neural networks for both PCA and quantization, this goal could e.g. be achieved by a modular network as in Figure 3 for the 4/2 pyramid. For quantization, we could either apply a vector quantizer to a whole patch of the Laplace level, or use a scalar quantizer (as depicted in Figure 3) for each pixel of the Laplace level. In the second case, we have to constrain the weights of the quantization network to be identical for every Laplace pixel. Since scalar quantization is simpler to analyze and uses less free parameters, we only consider this case.

As each quantization subnetwork can be treated separately (we only have to average the weight changes over all subnetworks), the following only considers the case of one output unit of the PCA network.

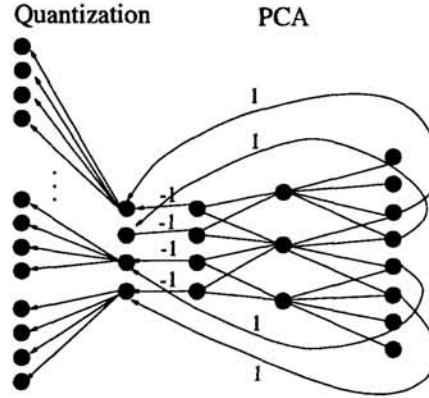

Figure 3: PCA network and Quantization network

The error to be minimized is the squared quantization error

$$E = \sum_p E_p \qquad\qquad E_p = \min_k \|c_k - l_p\|^2,$$

where $p$ refers to the patterns in the training set, $c_k$ is the $k$th weight of the quantization network, and $l$ is the output of the PCA-Laplace unit.

Changing the weights of the quantization network by gradient descent leads to the LVQ1 rule of Kohonen

$$\Delta c_k = \begin{cases} 2\alpha(l_p - c_k), & \text{if } k = k_p \text{ is the winning unit,} \\ 0, & \text{otherwise.} \end{cases}$$

For the PCA network we can proceed similarly to back-propagation to obtain the rule

$$\Delta w_{ij} = -K\frac{\partial E_p}{\partial w_{ij}} = -K\frac{\partial E_p}{\partial l_p}\frac{\partial l_p}{\partial w_{ij}} = -K\frac{\partial E_p}{\partial l_p}\frac{\partial l_p}{\partial i'_p}\frac{\partial i'_p}{\partial w_{ij}} = -2K(l_p - c_k)\frac{\partial i'_p}{\partial w_{ij}}.$$

Of course, this is only one out of many possible algorithms. More elaborate minimization techniques than gradient descent could be used; similarly, LVQ1 could be replaced by a different quantization algorithm. But the basic idea of letting the quantization step and the the compression step adapt to each other remains unchanged.

## 4  Conclusions

In this paper, we presented a new image compression scheme based on neural networks. The PCA and PCA-Laplace pyramids were introduced, which can be seen as both an extension of image pyramids to learning pyramids and as cascaded, locally connected autoassociators. The results achieved are promising and compare favorably to work reported in the literature.

A lot of work remains to be done to analyze these networks analytically. The convergence properties of the PCA pyramid are not known; we expect results similar

to the ones (Baldi and Hornik, 1989) for the autoassociative network. Also, for the PCA network it would be desirable to characterize the features which are extracted. Similarly, the integrated network needs to be analyzed. It is clear that for such networks, the usual error function has local minima, but maybe they can be avoided by a proper training regime (i.e. start training the PCA pyramid, then train the vector quantizer, and finally train them together).

## Footnotes

[1]In all our experiments the training algorithm converged (i.e. usually after 200 steps, however the improvements between steps 20 and convergence are negligible).

## References

Baldi, P. and Hornik, K. (1989). Neural Networks and principal component analysis: Learning from examples without local minima. *Neural Networks*, 2:53–58.

Baldi, P. and Hornik, K. (1995). Learning in Linear Neural Networks: a Survey. *IEEE Transactions on Neural Networks, to appear.*

Bischof, H. (1993). *Pyramidal Neural Networks*. PhD thesis, TU-Vienna, Inst. f. Automation, Dept. f. Pattern Recognition and Image Processing.

Bourlard, H. and Kamp, Y. (1988). Auto-Association by Multilayer Perceptrons and Singular Value Decomposition. *Biological Cybernetics*, 59:291–294.

Burt, P. J. and Adelson, E. H. (1983). The Laplacian pyramid as a compact image code. *IEEE Transactions on Communications*, Vol. COM-31(No.4):pp.532–540.

Cottrell, G., Munro, P., and Zipser, D. (1987). Learning Internal Representations from Grey-Scale Images: An Example of Extensional Programming. In *Ninth Annual Conference of the Cognitive Science Society*, pages 462–473. Hillsdale Erlbaum.

Hornik, K. and Kuan, C. (1992). Convergence analysis of local feature extraction algorithms. *Neural Networks*, 5(2):229–240.

Mallat, S. G. (1989). A Theory for Multiresolution Signal Decomposition: The Wavelet Representation. *IEEE Transactions on Pattern Analysis and Machine Intelligence*, Vol. PAMI-11(No. 7):pp. 674–693.

Mayer, H. and Kropatsch, W. G. (1989). Progressive Bildübertragung mit der $3\times3/2$ Pyramide. In Burkhardt, H., Höhne, K., and Neumann, B., editors, *Informatik Fachberichte 219: Mustererkennung 1989*, pages 160–167, Hamburg. 11.DAGM - Symposium, Springer Verlag.

Mougeot, M., Azencott, R., and Angeniol, B. (1991). Image Compression with Back Propagation: Improvement of the Visual Restoration using different Cost Functions. *Neural Networks*, 4:467–476.

Sanger, T. (1989). Optimal Unsupervised learning in a Single-Layer Linear Feed-forward Neural Network. *Neural Networks*, 2:433–459.

Schweizer, L., Parladori, G., Sicranza, G., and Marsi, S. (1991). A fully neural approach to image compression. In Kohonen, T., Mäkissara, K., Simula, O., and Kangas, J., editors, *Artificial Neural Networks*, volume I, pages 815–820.

Ziv, J. and Lempel, A. (1977). A universal algorithm for sequential data compression. *IEEE Trans. on Information Theory*, 23(5):337 – 343.